# The Use of MDL to Select among Computational Models of Cognition

**In J. Myung, Mark A. Pitt & Shaobo Zhang**
Department of Psychology
Ohio State University
Columbus, OH 43210
*{myung.1, pitt.2}@osu.edu*

**Vijay Balasubramanian**
David Rittenhouse Laboratories
University of Pennsylvania
Philadelphia, PA 19103
*vijay@endiv.hep.upenn.edu*

## Abstract

How should we decide among competing explanations of a cognitive process given limited observations? The problem of model selection is at the heart of progress in cognitive science. In this paper, Minimum Description Length (MDL) is introduced as a method for selecting among computational models of cognition. We also show that differential geometry provides an intuitive understanding of what drives model selection in MDL. Finally, adequacy of MDL is demonstrated in two areas of cognitive modeling.

## 1 Model Selection and Model Complexity

The development and testing of computational models of cognitive processing are a central focus in cognitive science. A model embodies a solution to a problem whose adequacy is evaluated by its ability to mimic behavior by capturing the regularities underlying observed data. This enterprise of model selection is challenging because of the competing goals that must be satisfied. Traditionally, computational models of cognition have been compared using one of many goodness-of-fit measures. However, use of such a measure can result in the choice of a model that over-fits the data, one that captures idiosyncrasies in the particular data set (i.e., noise) over and above the underlying regularities of interest. Such models are considered complex, in that the inherent flexibility in the model enables it to fit diverse patterns of data. As a group, they can be characterized as having many parameters that are combined in a highly nonlinear fashion in the model equation. They do not assume a single structure in the data. Rather, the model contains multiple structures; each obtained by finely tuning the parameter values of the model, and thus can fit a wide range of data patterns. In contrast, simple models, frequently with few parameters, assume a specific structure in the data, which will manifest itself as a narrow range of similar data patterns. Only when one of these patterns occurs will the model fit the data well.

The problem of over-fitting data due to model complexity suggests that the goal of model selection should instead be to select the model that generalizes best to all data samples that arise from the same underlying regularity, thus capturing only the regularity, not the noise. To achieve this goal, the selection method must be sensitive to the complexity of a model. There are at least two independent dimensions of model complexity. They are the number of free parameters of a

model and its functional form, which refers to the way the parameters are combined in the model equation. For instance, it seems unlikely that two one-parameter models, $y = \theta x$ and $y = x^{\theta}$, are equally complex in their ability to fit data. The two dimensions of model complexity (number of parameters and functional form) and their interplay can improve a model's fit to the data, without necessarily improving generalizability.

The trademark of a good model selection procedure, then, is its ability to satisfy two opposing goals. A model must be sufficiently complex to describe the data sample accurately, but without over-fitting the data and thus losing generalizability. To achieve this end, we need a theoretically well-justified measure of model complexity that takes into account the number of parameters and the functional form of a model. In this paper, we introduce Minimum Description Length (MDL) as an appropriate method of selecting among mathematical models of cognition. We also show that MDL has an elegant geometric interpretation that provides a clear, intuitive understanding of the meaning of complexity in MDL. Finally, application examples of MDL are presented in two areas of cognitive modeling.

## 1.1 Minimum Description Length

The central thesis of model selection is the estimation of a model's generalizability. One approach to assessing generalizability is the Minimum Description Length (MDL) principle [1]. It provides a theoretically well-grounded measure of complexity that is sensitive to both dimensions of complexity and also lends itself to intuitive, geometric interpretations. MDL was developed within algorithmic coding theory to choose the model that permits the greatest compression of data. A model family $f$ with parameters $\theta$ assigns the likelihood $f(y|\theta)$ to a given set of observed data $y$. The full form of the MDL measure for such a model family is given below.

$$MDL = -\ln f(y|\hat{\theta}) + \frac{k}{2}\ln\left(\frac{N}{2\pi}\right) + \ln \int d\theta \sqrt{\det I(\theta)}$$

where $\hat{\theta}$ is the parameter that maximizes the likelihood, $k$ is the number of parameters in the model, $N$ is the sample size and $I(\theta)$ is the Fisher information matrix. MDL is the length in bits of the shortest possible code that describes the data with the help of a model. In the context of cognitive modeling, the model that minimizes MDL uncovers the greatest amount of regularity (i.e., knowledge) underlying the data and therefore should be selected. The first, maximized log likelihood term is the lack-of-fit measure, and the second and third terms constitute the intrinsic complexity of the model. In particular, the third term captures the effects of complexity due to functional form, reflected through $I(\theta)$. We will call the latter two terms together the *geometric complexity* of the model, for reasons that will become clear in the remainder of this paper.

MDL arises as a finite series of terms in an asymptotic expansion of the Bayesian posterior probability of a model given the data for a special form of the parameter prior density [2]. Hence in essence, minimization of MDL is equivalent to maximization of the Bayesian posterior probability. In this paper we present a geometric interpretation of MDL, as well as Bayesian model selection [3], that provides an elegant and intuitive framework for understanding model complexity, a central concept in model selection.

## 2 Differential Geometric Interpretation of MDL

From a geometric perspective, a parametric model family of probability distributions forms a Riemannian manifold embedded in the space of all probability

distributions [4]. Every distribution is a point in this space, and the collection of points created by varying the parameters of the model gives rise to a hyper-surface in which ``similar'' distributions are mapped to ``nearby'' points. The infinitesimal distance between points separated by the infinitesimal parameter differences $d\theta^i$ is given by $ds^2 = \sum_{i,j=1}^{k} g_{ij}(\theta)d\theta^i d\theta^j$ where $g_{ij}(\theta)$ is the Riemannian metric tensor. The Fisher information, $I_{ij}(\theta)$, is the natural metric on a manifold of distributions in the context of statistical inference [4]. We argue that the MDL measure of model fitness has an attractive interpretation in such a geometric context.

The first term in MDL estimates the accuracy of the model since the likelihood $f(y|\hat{\theta})$ measures the ability of the model to fit the observed data. The second and third terms are supposed to penalize model complexity; we will show that they have interesting geometric interpretations. Given the metric $g_{ij} = I_{ij}$ on the space of parameters, the infinitesimal volume element on the parameter manifold is $dV = d\theta\sqrt{\det I(\theta)} \equiv \prod_{i=1}^{k} d\theta^i \sqrt{\det I(\theta)}$. The Riemannian volume of the parameter manifold is obtained by integrating $dV$ over the space of parameters:

$$V_M = \int dV = \int d\theta \sqrt{\det I(\theta)}$$

In other words, the third term in MDL penalizes models that occupy a large volume in the space of distributions.

In fact, the volume measure $V_M$ is related to the number of "distinguishable" probability distributions indexed by the model M.[1] Because of the way the model family is embedded in the space of distributions, two different parameter values can index very similar distributions. If complexity is related to volumes occupied by model manifolds, the measure of volume should count only different, or distinguishable, distributions, and not the artificial coordinate volume. It is shown in [2,5] that the volume $V_M$ achieves this goal.[2]

While the third term in MDL measures the total volume of distributions a model can describe, the second term relates to the number of model distributions that lie close to the truth. To see this, taking a Bayesian perspective on model selection is helpful. Using Bayes rule, the probability that the truth lies in the family $f$ given the observed data $y$ can be written as:

$$\Pr(f|y) = A(f,y)\int d\theta\, w(\theta)\Pr(y|\theta)$$

Here $w(\theta)$ is the prior probability of the parameter $\theta$, and $A(f, y) = \Pr(f)/\Pr(y)$ is the ratio of the prior probabilities of the family $f$ and data $y$. Bayesian methods assume that the latter are the same for all models under consideration and analyze the so-called Bayesian posterior

$$P_f = \int d\theta\, w(\theta)\Pr(y|\theta)\cdot$$

Lacking prior knowledge, $w$ should be chosen to weight all distinguishable distributions in the family equally. Hence, $w(\theta) = 1/V_M$. For large sample sizes, the likelihood function $f(y|\hat{\theta})$ localizes under general conditions to a multivariate

Gaussian centered at the maximum likelihood parameter $\hat{\theta}$ (see [3,4] and citations therein). In this limit, the integral for $P_f$ can be explicitly carried out. Performing the integral and taking a log given the result

$$- \ln P_f = - \ln f(y|\hat{\theta}) + \ln(V_M / C_M) + O(1/N) \quad where \ C_M = (2\pi/N)^{k/2} h(\hat{\theta})$$

where $h(\hat{\theta})$ is a data-dependent factor that goes to 1 for large $N$ when the truth lies within $f$ (see [3,4] for details). $C_M$ is essentially the volume of an ellipsoidal region around the Gaussian peak at $f(y|\hat{\theta})$ where the integrand of the Bayesian posterior makes a substantial contribution. In effect, $C_M$ measures the number of distinguishable distributions within $f$ that lie close to the truth.

Using the expressions for $C_M$ and $V_M$, the MDL selection criterion can be written as

$$MDL = - \ln f(y|\hat{\theta}) + \ln(V_M / C_M) + terms\ subleading\ in\ N$$

(The subleading terms include the contribution of $h(\hat{\theta})$; see [3,4] regarding its role in Bayesian inference.) The geometric meaning of the complexity penalty in MDL now becomes clear; models which occupy a relatively large volume distant from the truth are penalized. Models that contain a relatively large fraction of distributions lying close to the truth are preferred. Therefore, we refer to the last two terms in MDL as *geometric complexity*. It is also illuminating to collect terms in MD as

$$MDL = - \ln\left(\frac{f(y|\hat{\theta})}{(V_M / C_M)}\right) = - \ln("normalized\ \max imized\ likelihood")$$

Written this way, MDL selects the model that gives the highest value of the maximum likelihood, per the relative ratio of distinguishable distributions $(V_M/C_M)$. From this perspective, a better model is simply one with many distinguishable distributions close to the truth, but few distinguishable distributions overall.

# 3 Application Examples

Geometric complexity and MDL constitute a powerful pair of model evaluation tools. When used together in model testing, a deeper understanding of the relationship between models can be gained. The first measure enables one to assess the relative complexities of the set of models under consideration. The second builds on the first by suggesting which model is preferable given the data in hand. The following simulations demonstrate the application of these methods in two areas of cognitive modeling: information integration, and categorization. In each example, two competing models were fitted to artificial data sets generated by each model. Of interest is the ability of a selection method to recover the model that generated the data. MDL is compared with two other selection methods, both of which consider the number of parameters only. They are the Akaike Information Criterion (AIC; [6]) and the Bayesian Information Criterion (BIC; [7]) defined as:

$$AIC = -2\ln f(y|\hat{\theta}) + 2k; \ BIC = -2\ln f(y|\hat{\theta}) + k \ln N.$$

## 3.1 Information Integration

In a typical information integration experiment, a range of stimuli are generated from a factorial manipulation of two or more stimulus dimensions (e.g.,, visual and auditory) and then presented to participants for categorization as one of two or more possible response alternatives. Data are scored as the proportion of responses in one

category across the various combinations of stimulus dimensions. For this comparison, we consider two models of information integration, the Fuzzy Logical Model of Perception (FLMP; [8]) and the Linear Integration Model (LIM; [9]). Each assumes that the response probability ($p_{ij}$) of one category, say A, upon the presentation of a stimulus of the specific i and j feature dimensions in a two-factor information integration experiment takes the following form:

$$\textbf{FLMP}: p_{ij} = \frac{\theta_i \lambda_j}{\theta_i \lambda_j + (1 - \theta_i)(1 - \lambda_j)}; \ \textbf{LIM}: p_{ij} = \frac{\theta_i + \lambda_j}{2}$$

where $\theta_i$ and $\lambda_j$ ($i=1,..,q_1$; $j=1,..,q_2$; $0 < \theta_i, \lambda_j < 1$) are parameters representing the corresponding feature dimensions. The simulation results are shown in Table 1.

When the data were generated by FLMP, regardless of the selection method used, FLMP was recovered 100% of the time. This was true across all selection methods and across both sample sizes, except for MDL when sample size was 20. In this case, MDL did not perform quite as well as the other selection methods. When the data were generated by LIM, AIC or BIC fared much more poorly whereas MDL recovered the correct model (LIM) across both sample sizes. Specifically, under AIC or BIC, FLMP was selected over LIM half of the time for $N = 20$ (51% vs. 49%), though such errors were reduced for $N = 150$ (17% vs 83%).

Table 1: Model Recovery Rates for Two Information Integration Models

| Sample Size | Selection Method | Data from: Model fitted: | FLMP | LIM |
|---|---|---|---|---|
| N = 20 | AIC/BIC | FLMP | 100% | 51% |
|  |  | LIM | 0% | 49% |
|  | MDL | FLMP | 89% | 0% |
|  |  | LIM | 11% | 100% |
| N = 150 | AIC/BIC | FLMP | 100% | 17% |
|  |  | LIM | 0% | 83% |
|  | MDL | FLMP | 100% | 0% |
|  |  | LIM | 0% | 100% |

That FLMP is selected over LIM when a method such as AIC was used, even when the data were generated by LIM, suggests that FLMP is more complex than LIM. This observation was confirmed when the geometric complexity of each model was calculated. The difference in geometric complexity between FLMP and LIM was 8.74, meaning that for every distinguishable distribution for which LIM can account, FLMP can describe about $e^{8.74} \cong 6248$ distinguishable distributions. Obviously, this difference in complexity between the two models must be due to the functional form because they have the same number of parameters.

### 3.2 Categorization

Two models of categorization were considered in the present demonstration. They were the generalized context model (GCM: [10]) and the prototype model (PRT: [11]). Each model assumes that categorization responses follow a multinomial probability distribution with $p_{iJ}$ (probability of category $C_J$ response given stimulus $X_i$), which is given by

$$\text{GCM}: p_{iJ} = \frac{\sum_{j \in C_J} s_{ij}}{\sum_K \sum_{k \in C_K} s_{ik}}; \quad \text{PRT}: p_{iJ} = \frac{s_{iJ}}{\sum_K s_{iK}}$$

In the equation, $s_{ij}$ is a similarity measure between multidimensional stimuli $X_i$ and $X_j$, $s_{iJ}$ is a similarity measure between stimulus $X_i$ and the prototypic stimulus $X_J$ of category $C_J$. Similarity is measured using the Minkowski distance metric with the metric parameter r. The two models were fitted to data sets generated by each model using the six-dimensional scaling solution from Experiment 1 of [12] under the Euclidean distance metric of $r = 2$.

As shown in Table 2, under AIC or BIC, a relatively small bias toward choosing GCM was found using data generated from PRT when $N = 20$. When MDL was used to choose between the two models, there was improvement over AIC in correcting the bias. In the larger sample size condition, there was no difference in model recovery rate between AIC and MDL. This outcome contrasts with that of the preceding example, in which MDL was generally superior to the other selection methods when sample size was smallest.

Table 2: Model Recovery Rates for Two Categorization Models

| Sample Size | Selection Method | Data from: Model fitted: | GCM | PRT |
|---|---|---|---|---|
| N = 20 | AIC/BIC | GCM | 98% | 15% |
|  |  | PRT | 2% | 85% |
|  | MDL | GCM | 96% | 7% |
|  |  | PRT | 4% | 93% |
| N = 150 | AIC/BIC | GCM | 99% | 1% |
|  |  | PRT | 1% | 99% |
|  | MDL | GCM | 99% | 1% |
|  |  | PRT | 1% | 99% |

On the face of it, these findings would suggest that MDL is not much better than the other selection methods. After all, what else could cause this result? The only circumstances in which such an outcome is predicted under MDL is when the functional forms of the two models are similar (recall that the models have the same number of parameters), thus minimizing the differential contribution of functional form in the complexity term. Calculation of the geometric complexity of each model confirmed this suspicion. GCM is indeed only slightly more complex than PRT, the difference being equal to 0.60, so GCM can describe about two distributions ($e^{0.60} \cong 1.8$) for every distribution PRT can describe.

These simulation results together demonstrate usefulness of MDL and the geometric complexity measure in testing models of cognition. MDL's sensitivity to functional form was clearly demonstrated in its superior model recovery rate, especially when the complexities of the models differed by a nontrivial amount.

# 4 Conclusion

Model selection in cognitive science can proceed far more confidently with a clear understanding of why one model should be preferred over another. A geometric

interpretation of MDL helps to achieve this goal. The work carried out thus far indicates that MDL, along with the geometric complexity measure, holds considerable promise in evaluating computational models of cognition. MDL chooses the correct model most of the time, and geometric complexity provides a measure of how different the two models are in their capacity or power. Future work is directed toward extending this approach to other classes of models, such as connectionist networks.

## Acknowledgment and Authors Note

M.A.P. and I.J.M. were supported by NIMH Grant MH57472. V.B. was supported by the Society of Fellows and the Milton Fund of Harvard University, by NSF grant NSF-PHY-9802709 and by the DOE grant DOE-FG02-95ER40893. The present work is based in part on [5] and [13].

## Footnotes

[1] Roughly speaking, two probability distributions are considered indistinguishable if one is mistaken for the other even in the presence of an infinite amount of data. A careful definition of distinguishability involves use of the Kullback-Leibler distance between two probability distributions. For further details, see [3,4].

[2] Note that the parameters of the model are always assumed to be cut off in a manner to ensure that $V_M$ is finite.

## References

[1] Rissanen, J. (1996) Fisher information and stochastic complexity. *IEEE Transaction on Information Theory, 42*, 40-47.

[2] Balasubramanian, V. (1997) Statistical inference, Occam's razor and statistical mechanics on the space of probability distributions. *Neural Computation, 9*, 349-368.

[3] MacKay, D. J. C. (1992). Bayesian interpolation. *Neural Computation, 4*, 415-447.

[4] Amari, S. I. (1985) *Differential Geometrical Methods in Statistics*. Springer-Verlag.

[5] Myung, I. J., Balasubramanian, V., & Pitt, M. A. (1999) Counting probability distributions: Differential geometry and model selection. *Proceedings of the National Academy of Sciences USA, 97*, 11170-11175.

[6] Akaike, H. (1973) Information theory and an extension of the maximum likelihood principle, in B. N. Petrox and F. Caski, *Second international symposium on information theory*, pp. 267-281. Akademiai Kiado, Budapest.

[7] Schwarz, G. (1978) Estimating the dimension of a model. *The Annals of Statistics*, 6, 461-464.

[8] Oden, G. C., & Massaro, D. W. (1978) Integration of featural information in speech perception. *Psychological Review, 85*, 172-191.

[9] Anderson, N. H. (1981) *Foundations of Information Integration Theory*. Academic Press.

[10] Nosofsky, R. M. (1986) Attention, similarity and the identification-categorization relationship. *Journal of Experimental Psychology: General, 115*, 39-57.

[11] Reed, S. K. (1972) Pattern recognition and categorization. *Cognitive Psychology, 3*, 382-407.

[12] Shin, H. J., & Nosofsky, R. M. (1992) Similarity-scaling studies of dot-patten classification and recognition. *Journal of Experimental Psychology: General, 121*, 278-304.

[13] Pitt, M. A., Myung, I. J., & Zhang, S. (2000). Toward a method of selecting among computational models of cognition. *Submitted for publication*.
